# Hierarchies of adaptive experts

Michael I. Jordan          Robert A. Jacobs
Department of Brain and Cognitive Sciences
Massachusetts Institute of Technology
Cambridge, MA 02139

## Abstract

In this paper we present a neural network architecture that discovers a recursive decomposition of its input space. Based on a generalization of the modular architecture of Jacobs, Jordan, Nowlan, and Hinton (1991), the architecture uses competition among networks to recursively split the input space into nested regions and to learn separate associative mappings within each region. The learning algorithm is shown to perform gradient ascent in a log likelihood function that captures the architecture's hierarchical structure.

## 1  INTRODUCTION

Neural network learning architectures such as the multilayer perceptron and adaptive radial basis function (RBF) networks are a natural nonlinear generalization of classical statistical techniques such as linear regression, logistic regression and additive modeling. Another class of nonlinear algorithms, exemplified by CART (Breiman, Friedman, Olshen, & Stone, 1984) and MARS (Friedman, 1990), generalizes classical techniques by partitioning the training data into non-overlapping regions and fitting separate models in each of the regions. These two classes of algorithms extend linear techniques in essentially independent directions, thus it seems worthwhile to investigate algorithms that incorporate aspects of both approaches to model estimation. Such algorithms would be related to CART and MARS as multilayer neural networks are related to linear statistical techniques. In this paper we present a candidate for such an algorithm. The algorithm that we present partitions its training data in the manner of CART or MARS, but it does so in a parallel, on-line manner that can be described as the stochastic optimization of an appropriate cost functional.

Why is it sensible to partition the training data and to fit separate models within each of the partitions? Essentially this approach enhances the flexibility of the learner and allows the data to influence the choice between local and global representations. For example, if the data suggest a discontinuity in the function being approximated, then it may be more sensible to fit separate models on both sides of the discontinuity than to adapt a global model across the discontinuity. Similarly, if the data suggest a simple functional form in some region, then it may be more sensible to fit a global model in that region than to approximate the function locally with a large number of local models. Although global algorithms such as backpropagation and local algorithms such as adaptive RBF networks have some degree of flexibility in the tradeoff that they realize between global and local representation, they do not have the flexibility of adaptive partitioning schemes such as CART and MARS.

In a previous paper we presented a modular neural network architecture in which a number of "expert networks" compete to learn a set of training data (Jacobs, Jordan, Nowlan & Hinton, 1991). As a result of the competition, the architecture adaptively splits the input space into regions, and learns separate associative mappings within each region. The architecture that we discuss here is a generalization of the earlier work and arises from considering what would be an appropriate internal structure for the expert networks in the competing experts architecture. In our earlier work, the expert networks were multilayer perceptrons or radial basis function networks. If the arguments in support of data partitioning are valid, however, then they apply equally well to a region in the input space as they do to the entire input space, and therefore each expert should itself be composed of competing sub-experts. Thus we are led to consider recursively-defined hierarchies of adaptive experts.

## 2    THE ARCHITECTURE

Figure 1 shows two hierarchical levels of the architecture. (We restrict ourselves to two levels throughout the paper to simplify the exposition; the algorithm that we develop, however, generalizes readily to trees of arbitrary depth). The architecture has a number of *expert networks* that map from the input vector $\mathbf{x}$ to output vectors $\mathbf{y}_{ij}$. There are also a number of *gating networks* that define the hierarchical structure of the architecture. There is a gating network for each cluster of expert networks and a gating network that serves to combine the outputs of the clusters. The output of the $i^{\text{th}}$ cluster is given by

$$\mathbf{y}_i = \sum_j g_{j|i} \mathbf{y}_{ij} \tag{1}$$

where $g_{j|i}$ is the activation of the $j^{\text{th}}$ output unit of the gating network in the $i^{\text{th}}$ cluster. The output of the architecture as a whole is given by

$$\mathbf{y} = \sum_i g_i \mathbf{y}_i \tag{2}$$

where $g_i$ is the activation of the $i^{\text{th}}$ output unit of the top-level gating network.

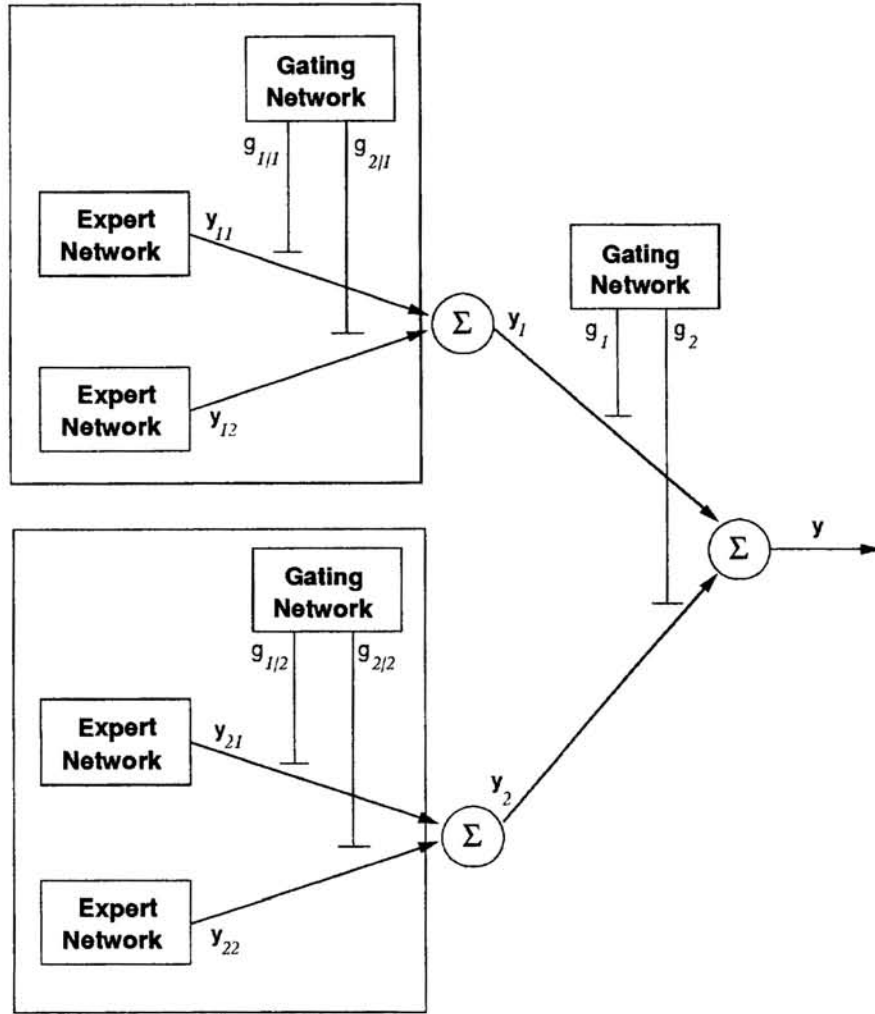

Figure 1: Two hierarchical levels of adaptive experts. All of the expert networks and all of the gating networks have the same input vector.

We assume that the outputs of the gating networks are given by the normalizing *softmax* function (Bridle, 1989):

$$g_i = \frac{e^{s_i}}{\sum_j e^{s_j}} \qquad (3)$$

and

$$g_{j|i} = \frac{e^{s_{j|i}}}{\sum_k e^{s_{k|i}}} \qquad (4)$$

where $s_i$ and $s_{j|i}$ are the weighted sums arriving at the output units of the corresponding gating networks.

The gating networks in the architecture are essentially classifiers that are responsible for partitioning the input space. Their choice of partition is based on the ability

of the expert networks to model the input-output functions within their respective regions (as quantified by their posterior probabilities; see below). The nested arrangement of gating networks in the architecture (cf. Figure 1) yields a nested partitioning much like that found in CART or MARS. The architecture is a more general mathematical object than a CART or MARS tree, however, given that the gating networks have non-binary outputs and given that they may form nonlinear decision surfaces.

## 3    THE LEARNING ALGORITHM

We derive a learning algorithm for our architecture by developing a probabilistic model of a tree-structured estimation problem. The environment is assumed to be characterized by a finite number of stochastic processes that map input vectors $\mathbf{x}$ into output vectors $\mathbf{y}^*$. These processes are partitioned into nested collections of processes that have commonalities in their input-output parameterizations. Data are assumed to be generated by the model in the following way. For any given $\mathbf{x}$, collection $i$ is chosen with probability $g_i$, and a particular process $j$ is then chosen with conditional probability $g_{j|i}$. The selected process produces an output vector $\mathbf{y}^*$ according to the probability density $f(\mathbf{y}^* \mid \mathbf{x}; \mathbf{y}_{ij})$, where $\mathbf{y}_{ij}$ is a vector of parameters. The total probability of generating $\mathbf{y}^*$ is:

$$P(\mathbf{y}^* \mid \mathbf{x}) = \sum_i g_i \sum_j g_{j|i} f(\mathbf{y}^* \mid \mathbf{x}; \mathbf{y}_{ij}), \tag{5}$$

where $g_i$, $g_{j|i}$, and $\mathbf{y}_{ij}$ are unknown nonlinear functions of $\mathbf{x}$.

Treating the probability $P(\mathbf{y}^* \mid \mathbf{x})$ as a likelihood function in the unknown parameters $g_i$, $g_{j|i}$, and $\mathbf{y}_{ij}$, we obtain a learning algorithm by using gradient ascent to maximize the log likelihood. Let us assume that the probability density associated with the residual vector $(\mathbf{y}^* - \mathbf{y}_{ij})$ is the multivariate normal density, where $\mathbf{y}_{ij}$ is the mean of the $j^{\text{th}}$ process of the $i^{\text{th}}$ cluster (or the $(i,j)^{\text{th}}$ expert network) and $\Sigma_{ij}$ is its covariance matrix. Ignoring the constant terms in the normal density, the log likelihood is:

$$\ln L = \ln \sum_i g_i \sum_j g_{j|i} |\Sigma_{ij}|^{-\frac{1}{2}} e^{-\frac{1}{2}(\mathbf{y}^* - \mathbf{y}_{ij})^T \Sigma_{ij}^{-1}(\mathbf{y}^* - \mathbf{y}_{ij})}. \tag{6}$$

We define the following posterior probability:

$$h_i = \frac{g_i \sum_j g_{j|i} |\Sigma_{ij}|^{-\frac{1}{2}} e^{-\frac{1}{2}(\mathbf{y}^* - \mathbf{y}_{ij})^T \Sigma_{ij}^{-1}(\mathbf{y}^* - \mathbf{y}_{ij})}}{\sum_i g_i \sum_j g_{j|i} |\Sigma_{ij}|^{-\frac{1}{2}} e^{-\frac{1}{2}(\mathbf{y}^* - \mathbf{y}_{ij})^T \Sigma_{ij}^{-1}(\mathbf{y}^* - \mathbf{y}_{ij})}}, \tag{7}$$

which is the posterior probability that a process in the $i^{\text{th}}$ cluster generates a particular target vector $\mathbf{y}^*$. We also define the conditional posterior probability:

$$h_{j|i} = \frac{g_{j|i} |\Sigma_{ij}|^{-\frac{1}{2}} e^{-\frac{1}{2}(\mathbf{y}^* - \mathbf{y}_{ij})^T \Sigma_{ij}^{-1}(\mathbf{y}^* - \mathbf{y}_{ij})}}{\sum_j g_{j|i} |\Sigma_{ij}|^{-\frac{1}{2}} e^{-\frac{1}{2}(\mathbf{y}^* - \mathbf{y}_{ij})^T \Sigma_{ij}^{-1}(\mathbf{y}^* - \mathbf{y}_{ij})}}, \tag{8}$$

which is the conditional posterior probability that the $j^{\text{th}}$ expert in the $i^{\text{th}}$ cluster generates a particular target vector $\mathbf{y}^*$. Differentiating 6, and using Equations 3, 4,

7, and 8, we obtain the partial derivative of the log likelihood with respect to the output of the $(i,j)^{\text{th}}$ expert network:

$$\frac{\partial \ln L}{\partial \mathbf{y}_{ij}} = h_i\, h_{j|i}\, (\mathbf{y}^* - \mathbf{y}_{ij}).\tag{9}$$

This partial derivative is a supervised error term modulated by the appropriate posterior probabilities. Similarly, the partial derivatives of the log likelihood with respect to the weighted sums at the output units of the gating networks are given by:

$$\frac{\partial \ln L}{\partial s_i} = h_i - g_i \tag{10}$$

and

$$\frac{\partial \ln L}{\partial s_{j|i}} = h_i\, (h_{j|i} - g_{j|i}).\tag{11}$$

These derivatives move the prior probabilities associated with the gating networks toward the corresponding posterior probabilities.

It is interesting to note that the posterior probability $h_i$ appears in the gradient for the experts in the $i^{\text{th}}$ cluster (Equation 9) and in the gradient for the gating network in the $i^{\text{th}}$ cluster (Equation 11). This ties experts within a cluster to each other and implies that experts within a cluster tend to learn similar mappings early in the training process. They differentiate later in training as the probabilities associated with the cluster to which they belong become larger. Thus the architecture tends to acquire coarse structure before acquiring fine structure. This feature of the architecture is significant because it implies a natural robustness to problems with overfitting in deep hierarchies.

We have also found it useful in practice to obtain an additional degree of control over the coarse-to-fine development of the algorithm. This is achieved with a heuristic that adjusts the learning rate at a given level of the tree as a function of the time-average entropy of the gating network at the next higher level of the tree:

$$\mu_{.|i}(t+1) = \alpha\mu_{.|i}(t) + \beta(M_i + \sum_j g_{j|i} \ln g_{j|i})$$

where $M_i$ is the maximum possible entropy at level $i$ of the tree. This equation has the effect that the networks at level $i+1$ are less inclined to diversify if the superordinate cluster at level $i$ has yet to diversify (where diversification is quantified by the entropy of the gating network).

# 4   SIMULATIONS

We present simulation results from an unsupervised learning task and two supervised learning tasks.

In the unsupervised learning task, the problem was to extract regularities from a set of measurements of leaf morphology. Two hundred examples of maple, poplar, oak, and birch leaves were generated from the data shown in Table 1. The architecture that we used had two hierarchical levels, two clusters of experts, and two experts

|         | Maple     | Poplar           | Oak       | Birch           |
|---------|-----------|------------------|-----------|-----------------|
| Length  | 3,4,5,6   | 1,2,3            | 5,6,7,8,9 | 2,3,4,5         |
| Width   | 3,4,5     | 1,2              | 2,3,4,5   | 1,2,3           |
| Flare   | 0         | 0,1              | 0         | 1               |
| Lobes   | 5         | 1                | 7,9       | 1               |
| Margin  | Entire    | Crenate, Serrate | Entire    | Doubly-Serrate  |
| Apex    | Acute     | Acute            | Rounded   | Acute           |
| Base    | Truncate  | Rounded          | Cumeate   | Rounded         |
| Color   | Light     | Yellow           | Light     | Dark            |

Table 1: Data used to generate examples of leaves from four types of trees. The columns correspond to the type of tree; the rows correspond to the features of a tree's leaf. The table's entries give the possible values for each feature for each type of leaf. See Preston (1976).

within each cluster. Each expert network was an auto-associator that maps forty-eight input units into forty-eight output units through a bottleneck of two hidden units. Within the experts, backpropagation was used to convert the derivatives in Equation 9 into changes to the weights. The gating networks at both levels were affine. We found that the hierarchical architecture consistently discovers the decomposition of the data that preserves the natural classes of tree species (cf. Preston, 1976). That is, within one cluster of expert networks, one expert learns the maple training patterns and the other expert learns the oak patterns. Within the other cluster, one expert learns the poplar patterns and the other expert learns the birch patterns. Moreover, due to the use of the autoassociator experts, the hidden unit representations within each expert are principal component decompositions that are specific to a particular species of leaf.

We have also studied a supervised learning problem in which the learner must predict the grayscale pixel values in noisy images of human faces based on values of the pixels in surrounding $5x5$ masks. There were 5000 masks in the training set. We used a four-level binary tree, with affine experts (each expert mapped from twenty-five input units to a single output unit) and affine gating networks. We compared the performance of the hierarchical architecture to CART and to backpropagation.[1] In the case of backpropagation and the hierarchical architecture, we utilized cross-validation (using a test set of 5000 masks) to stop the iterative training procedure. As shown in Figure 2, the performance of the hierarchical architecture is comparable to backpropagation and better than CART.

Finally we also studied a system identification problem involving learning the simulated forward dynamics of a four-joint, three-dimensional robot arm. The task was to predict the joint accelerations from the joint positions, sines and cosines of joint positions, joint velocities, and torques. There were 6000 data items in the training set. We used a four-level tree with trinary splits at the top two levels, and binary splits at lower levels. The tree had affine experts (each expert mapped

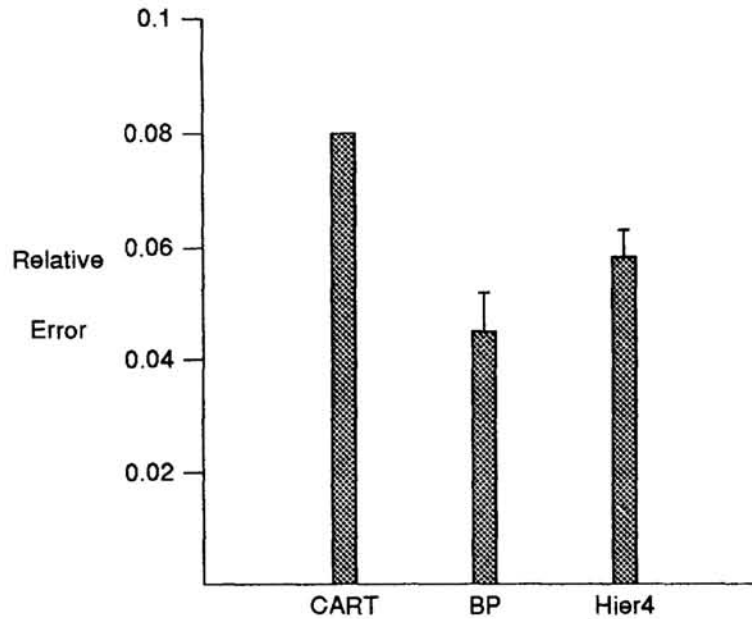

Figure 2: The results on the image restoration task. The dependent measure is relative error on the test set (cf. Breiman, et al., 1984).

from twenty input units to four output units) and affine gating networks. We once again compared the performance of the hierarchical architecture to CART and to backpropagation. In the case of backpropagation and the hierarchical architecture, we utilized a conjugate gradient technique, and halted the training process after 1000 iterations. In the case of CART, we ran the algorithm four separate times on the four output variables. Two of these runs produced 100 percent relative error, a third produced 75 percent relative error, and the fourth (the most proximal joint acceleration) yielded 46 percent relative error, which is the value we report in Figure 3. As shown in the figure, the hierarchical architecture and backpropagation achieve comparable levels of performance.

# 5   DISCUSSION

In this paper we have presented a neural network learning algorithm that captures aspects of the recursive approach to function approximation exemplified by algorithms such as CART and MARS. The results obtained thus far suggest that the algorithm is computationally viable, comparing favorably to backpropagation in terms of generalization performance on a set of small and medium-sized tasks. The algorithm also has a number of appealing theoretical properties when compared to backpropagation: In the affine case, it is possible to show that (1) no backward propagation of error terms is required to adjust parameters in multi-level trees (cf. the activation-dependence of the multiplicative terms in Equations 9 and 11), (2) all of the parameters in the tree are maximum likelihood estimators. The latter property suggests that the affine architecture may be a particularly suitable architecture in which to explore the effects of priors on the parameter space (cf. Nowlan

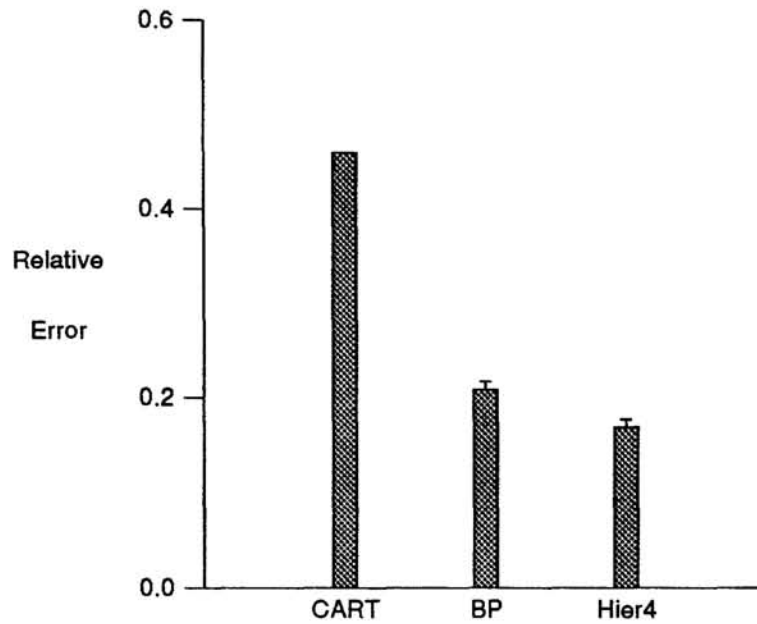

Figure 3: The results on the system identification task.

& Hinton, this volume).

## Acknowledgements

This project was supported by grant IRI-9013991 awarded by the National Science Foundation, by a grant from Siemens Corporation, by a grant from ATR Auditory and Visual Perception Research Laboratories, by a grant from the Human Frontier Science Program, and by an NSF Presidential Young Investigator Award to the first author.

## Footnotes

[1]Fifty hidden units were used in the backpropagation network, making the number of parameters in the backpropagation network and the hierarchical network roughly comparable.

## References

Breiman, L., Friedman, J.H., Olshen, R.A., & Stone, C.J. (1984) *Classification and Regression Trees*. Belmont, CA: Wadsworth International Group.

Bridle, J. (1989) Probabilistic interpretation of feedforward classification network outputs, with relationships to statistical pattern recognition. In F. Fogelman–Soulie & J. Hérault (Eds.), *Neuro-computing: Algorithms, Architectures, and Applications*. New York: Springer–Verlag.

Friedman, J.H. (1990) Multivariate adaptive regression splines. *The Annals of Statistics, 19*, 1–141.

Jacobs, R.A, Jordan, M.I., Nowlan, S.J., & Hinton, G.E. (1991) Adaptive mixtures of local experts. *Neural Computation, 3*, 79–87.

Preston, R.J. (1976) *North American Trees (Third Edition)*. Ames, IA: Iowa State University Press.